# Adaptive Object Representation with Hierarchically-Distributed Memory Sites

**Bosco S. Tjan**
Department of Psychology
University of Southern California
*btjan@usc.edu*

## Abstract

Theories of object recognition often assume that only one representation scheme is used within one visual-processing pathway. Versatility of the visual system comes from having multiple visual-processing pathways, each specialized in a different category of objects. We propose a theoretically simpler alternative, capable of explaining the same set of data and more. A single primary visual-processing pathway, loosely modular, is assumed. Memory modules are attached to sites along this pathway. Object-identity decision is made independently at each site. A site's response time is a monotonic-decreasing function of its confidence regarding its decision. An observer's response is the first-arriving response from any site. The effective representation(s) of such a system, determined empirically, can appear to be specialized for different tasks and stimuli, consistent with recent clinical and functional-imaging findings. This, however, merely reflects a decision being made at its appropriate level of abstraction. The system itself is intrinsically flexible and adaptive.

## 1 Introduction

How does the visual system represent its knowledge about objects so as to identify them? A largely unquestioned assumption in the study of object recognition has been that the visual system builds up a representation for an object by sequentially transforming an input image into progressively more abstract representations. The final representation is taken to be *the* representation of an object and is entered into memory. Recognition of an object occurs when the representation of the object currently in view matches an item in memory.

Highly influential proposals for a common representation of objects [1, 2] have failed to show promise of either producing a working artificial system or explaining a gamut of behavioral data. This insistence of having a common representation for all objects is also a major cause of the debate on whether *the* perceptual representation of objects is 2-D appearance-based or 3-D structure-based [3, 4].

Recently, a convergence of data [5-9], including those from the viewpoint debate itself [10, 11], have been used to suggest that the brain may use multiple

mechanisms or processing pathways to recognize a multitude of objects. While insisting on a common representation for all objects seems too restrictive in light of the varying complexity across objects [12], asserting a new pathway for every idiosyncratic data clusters seems unnecessary.

We propose a parsimonious alternative, which is consistent with existing data but explains them with novel insights. Our framework relies on a single processing pathway. Flexibility and self-adaptivity are achieved by having multiple memory and decision sites distributed along the pathway.

## 2  Theory and Methods

If the visual system needs to construct an abstract representation of objects for a certain task (e.g. object categorization), it will have to do so via multiple stages. The intermediate result at each stage is itself a representation. The entire processing pathway thus provides a hierarchy of representations, ranging from the most image-specific at the earliest stage to the most abstract at the latest stage.

The central idea of our proposal is that the visual system can tap this hierarchical collection of representations by attaching memory modules along the processing pathway. We further speculate that each memory site makes independent decisions about the identity of an incoming image. Each announces its decision after a delay, determined by an amount related to the site's confidence about its own decision and the amount of memory it needs to consult before reaching the decision. The homunculus does nothing but takes the first-arriving response as the system's response. Figure 1a depicts this framework, which we shall call the Hierarchically Distributed Decision Theory for object recognition.

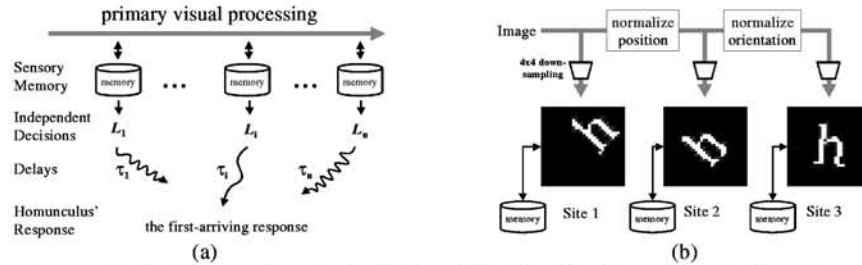

**Figure 1**: An illustration of the Hierarchically Distributed Decision Theory of object recognition (a) and its implementation in a toy visual system (b).

### 2.1  A toy visual system

We constructed a toy visual system to illustrate various properties of the Hierarchically Distributed Decision Theory. The task for this toy system is to identify letters presented at arbitrary position and orientation and corrupted by Gaussian luminance noise. This system is not meant to be a model of human vision, but rather a demonstration of the theory. Given a stimulus (letter+noise), the position of the target letter is first estimated and centered in the image (position normalization) by computing the centroid of the stimulus' luminance profile. Once centered, the principal axis of the luminance profile is determined and the entire image is rotated so that this axis is vertical (orientation normalization). The representation at this final stage is both position- and orientation-invariant. Traditionally, one would commit only this final representation to memory.

In contrast, the Hierarchically Distributed Decision Theory stated that the intermediate results are also committed to some form of sensory memory (Figure

1b). A memory item is a feature vector. For this toy system, a feature vector is a sub-sampled image at the output of each stage. To recognize a letter, each site $s$ independently decides the letter's identity $L_s$, based on the immediate representation $I_s$ available to the site. It does so by maximizing the posterior probability $\Pr(L_s|I_s)$, assuming 1) independent feature noise of known distribution (in this case, independent Gaussian luminance noise of zero mean and standard deviation $\sigma$) and 2) that its memory content completely captures all other sources of correlated noise and signal uncertainties (deviation from which is assessed by Eq. 3). Specifically,

$$L_s = \arg \max_{r \in Letters} \Pr(r \mid I_s) \tag{1}$$

where *Letters* is the set of letter identities. A letter identity $r$ is in turn a set of letter images $V$ at a given luminance, which may be shifted or rotated. So we have,

$$\Pr(r|I_s) = \sum_{V \in r} \Pr(V \mid I_s) = \sum_{V \in r} \Pr(I_s \mid V) \Pr(V) \Big/ \Pr(I_s)$$

$$= \sum_{V \in r} \exp\left(\frac{-\|I_s - V\|^2}{2s^2}\right) \Pr(V) \Big/ \sum_{r \in Letters} \sum_{V \in r} \exp\left(\frac{-\|I_s - V\|^2}{2s^2}\right) \Pr(V) \tag{2}$$

In addition to choosing a response, each site delays sending out its response by an amount $\tau_s$. $\tau_s$ is related to each site's own assessment of its confidence about its decision and the size of memory it needed to consult to make the decision. $\tau_s$ is a monotonically decreasing function of confidence (one minus the maximum posterior probability) and a monotonically increasing function of memory size:

$$\tau_s = h_1 \sqrt{1 - \max_{r \in Letters} \Pr(r \mid I_s)} + h_2 \log(M_s) + h_0 \tag{3}$$

$h_0$, $h_1$, and $h_2$, are constants common to all sites. $M_s$ is the effective number of items in memory at site $s$, equal to the number of distinct training views the site saw (or the limit of its memory size, whichever is less). In our toy system, $M_1$ is the number of distinct training views presented to the system. $M_2$ is approximately the number of training views with distinct orientations (because $I_2$ is normalized by position), and $M_3$ is effectively one view per letter. In general, $M_1 > M_2 > M_3$. Relative to the *decision* time $\tau_s$, the *processing* time required to perform normalizations is assumed to be negligible (This assumption can be removed by letting $h_0$ depend on site $s$.)

## 2.2   Learning and testing

The learning component of the theory has yet to be determined. For our toy system, we assumed that the items kept in memory are free of luminance noise but subjected to normalization errors caused by the luminance noise (e.g. the position of a letter may not be perfectly determined).

We measured performance of the toy system by first exposing it to 5 orientations and 20 positions of each letter at high signal-to-noise ratio (SNR). Ten letters from the Times Roman font were used in the simulation (bcdeghnopw). The system keeps in memory those studied views (Site 1) and their normalized versions (Sites 2 & 3). Therefore, $M_1 = 5 \times 20 \times 10 = 1000$. Since the normalization processes are reliable at high SNR, $M_2 \approx 50$, and $M_3 \approx 10$.

We tested the system by presenting it with letters from either the studied views, or views it had not seen before. In the latter case, a novel view could be either with novel position alone, or with both novel position and orientation. The test stimuli were presented at SNR ranging from 210 to 1800 (Weber contrast of 10-30% at mean luminance of 48 cd/m$^2$ and a noise standard deviation of 10 cd/m$^2$).

# 3 Results and Discussions

Figure 2a shows the performance of our toy visual system under different stimulus conditions. The numbered thin curves indicate recognition accuracy achieved by each site. As expected, Site 1, which kept raw images in memory, achieved the best accuracy when tested with studied views, but it could not generalize to novel views. In contrast, Site 3 maintained essentially the same level of performance regardless of view condition – its representation was invariant to position and orientation.

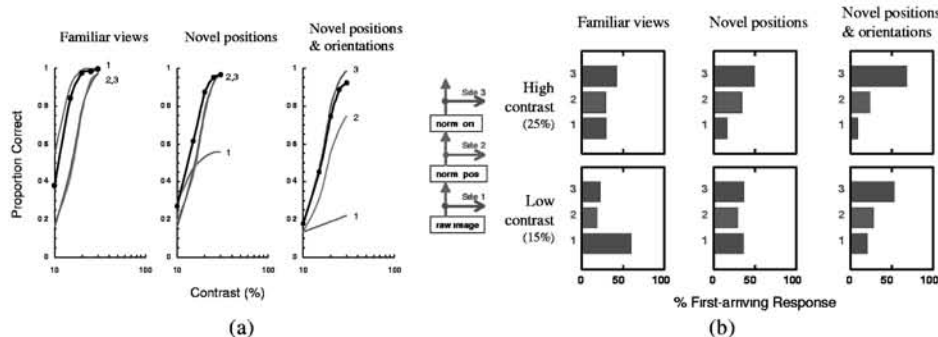

**Figure 2**: (a) Accuracy of the system (solid symbols) verses accuracy of each site (numbered curves) under different contrast and view conditions. (b) Relative frequency of a site issuing the first-arriving response.

The thick curves with solid symbols indicated the system's performance based on first-arriving responses. Clearly, it tracked the performance of the best-performing site under all conditions. Under the novel-position condition, the system's performance was even better than the best-performing sites. This is because although Site 2 and 3 performed equally well, they made different errors. The simple delay rule effectively picked out the most reliable response at each trial.

Figure 2b shows the source distribution of the first-arriving responses. When familiar (i.e. studied) views were presented at low contrast (low SNR), Site 1, which used raw image as the representation, was responsible for issuing about 60% of the first-arriving responses. This is because normalization processes tend to be less reliable at low SNR. Whenever an input to Site 2 or 3 cannot be properly normalized, it will match poorly to the normalized views in memory, resulting in lower confidence and longer delay. As contrast increased, normalization processes became more accurate, and the first-arriving responses shifted to the higher sites. Higher sites encode more invariance, and thus need to consult fewer memory items. Lastly, when novel views were presented, Site 3 tended to be the most active, since it was the only site that fully captured all the invariance necessary for this condition.

The delay mechanism specified by Eq. 3 allows the system as a whole to be self-adaptive. Its effective representation, if we can speak of such, is flexible. No site is exclusively responsible for any particular kind of stimuli. Instead, the decision is always distributed across sites in a trial-by-trial basis.

What do existing human data on object recognition have to say about this simple framework? Wouldn't those data supporting functional specialization or object-category-specific representations argue against this framework? Not at all!

## 3.1 Viewpoint effects

Entry-level object recognition [13] often shows less viewpoint dependence than subordinate-level object recognition. This has been taken to suggest that two

different mechanisms or forms of representation may be subserving these two types of object recognition tasks [4].

Figure 3a shows our system's overall performance in response time (RT) and error rate when tested with the studied (thus "familiar") and the novel (new positions and orientations) views. The difference in RT and error rate between these two conditions (Figure 3b) is a rough measure of the viewpoint effect. Even though the system includes a site (Site 3) with viewpoint-invariant representation, the system's overall performance still depends on viewpoint, particularly at low contrast.

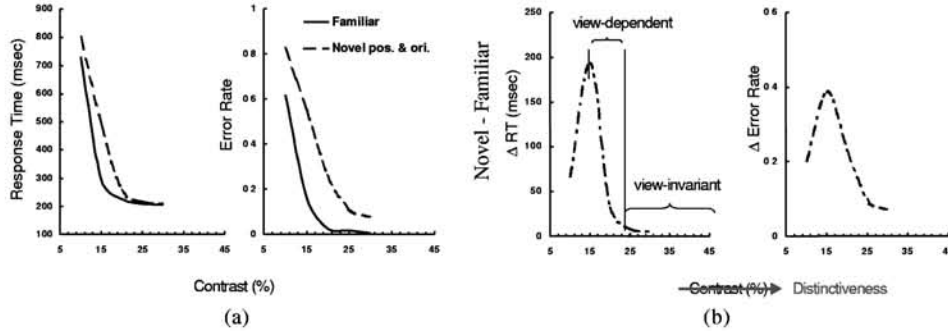

**Figure 3**: (a) RT and error rate of the toy system when tested with either the studied or novel views. (b) Difference between the two conditions.

Because the representation space of this toy system is the image space, contrast is a direct measure of "perceptual" distinctiveness. Figure 3b shows that when objects were sufficiently distinct (as in entry-level recognition), there was little or no viewpoint effect. When objects were highly similar, performances were equally poor for studied and novel views, so there was little viewpoint effect to speak of. Viewpoint effect was localized to a mid-range of distinctiveness. Within this range, increasing similarity increased viewpoint dependence. The fact that viewpoint effect was present only within a bounded range of distinctiveness agrees with the general experience that sizable viewpoint effect is uncommon unless artificially-created objects or objects chosen from the same category (subordinate-level recognition) are used.

## 3.2 Functionally specialized brain regions

Various fMRI studies have observed what appears to be functionally specialized brain regions involved in object perception [7-9]. To identify and localize such areas, a typical approach is to subtract the observed hemodynamic signals under one stimulus condition from that under a different condition. An area is said to be "selective" to a stimulus type X if its signal strength is higher whenever X, as opposed to some other type of stimuli, is displayed.

We performed a simulated "imaging" on our toy visual system. Consider Figure 3b. If we assume that one unit of metabolic energy is needed to send a response, and no more response will be sent after the first-arriving response has been received, we can re-label the x-axis of the histograms as "hemodynamic signal", or "activation level". Furthermore, as mentioned before, we can label stimuli in high contrast as "distinct objects" and those in low contrast as "similar objects."

When we did "similar minus distinct", we obtained the result shown in the lower right-hand panel in Figure 4a. Site 1 was more active than all other sites when recognition was between similar objects, while Site 3 was more active when recognition was between distinct objects. The standard practice of interpreting such a result would label Site 1 as an area for processing similar (perhaps subordinate-

level) objects, and Site 3 as an area for processing distinct (perhaps entry-level) objects. Knowing how the decisions are actually made however, such labeling is clearly misguided.

When instead we did "familiar minus novel", we obtained a similar pattern of result (Figure 4a, upper right). However, this time we would have to label Site 1 as an area for processing familiar objects (or an area for expertise), and Site 3 for novel objects. Analogous to an on-going debate about expertise vs. object-specificity [14], whether Site 1 is for familiar objects or similar objects cannot be resolved based on subtraction method alone.

According to the standard interpretation of the subtraction method, our toy visual system *appeared* to contain functionally specialized sites; yet, none of the sites were designed to specialize in any kind of stimuli. Even in the most extreme cases, no site was responsible for more than 70% of the decisions.

One last point is worth mentioning. The primary visual pathway was equally active under all conditions, so its activity became invisible after subtraction. The observed signal change revealed only the difference in memory activities.

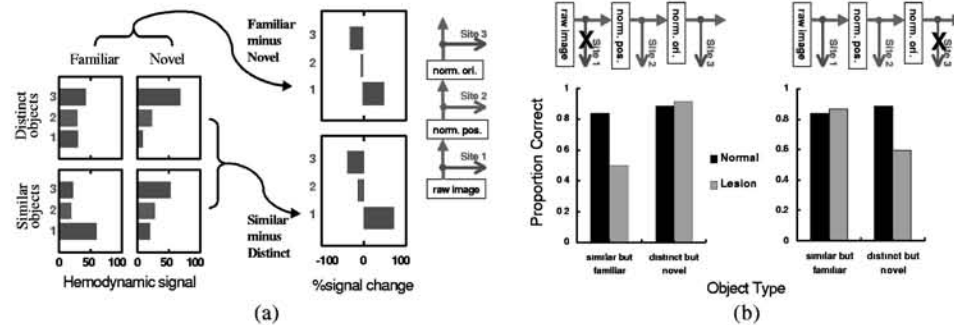

**Figure 4**: The toy visual system gives the appearance of containing functionally specialized modules in simulated functional imaging (a) and lesion studies (b).

### 3.3 Category-specific deficits

Patients with prosopagnosia cannot recognize faces, but their ability for recognizing other objects are often spared. Patients with visual object agnosia have the opposite impairments. This kind of double dissociation is taken as another evidence to suggest that the visual system contains object-specific modules (cf. [15]).

We observed the same kind of double dissociation with our toy model. Figure 4b shows what happened when we "lesioned" different memory sites in our system by preventing a site from issuing any response. When Site 1 was lesioned, recognition performance for similar-but-familiar objects (analog to familiar faces) was impeded while performance for distinct-but-novel objects was spared. The opposite was true when Site 3 was lesioned. It is worth restating that our toy system consisted of only a single processing pathway and no category-specific representations.

## 4 Conclusion

Intermediate representations along a single visual-processing pathway form a natural hierarchy of abstractions. We have shown that by attaching sensory memory modules to the pathway, this hierarchy can be exploited to achieve an effective representation of objects that is highly flexible and adaptive. Each memory module

makes independent decision regarding the identity of an object based on the intermediate representation available to it. Each module delays sending out its response by an amount related to its confidence about its decision, in addition to the time required for memory lookup. The first-arriving response becomes the system's response.

It is an attractive conjecture that this scheme of adaptive representation may be used by the visual system. Through a toy example, we have shown that such a system can appear to behave like one with multiple functionally specialized pathways or category-specific representations, raising questions for the contemporary interpretations of behavioral, clinical and functional-imaging data regarding the neuro-architecture for object recognition.

## References

1. Marr, D., *Vision*. 1982, San Francisco: Freeman.
2. Biederman, I., *Recognition-by-components: A theory of human image understanding*. Psychological Review, 1987. **94**: p. 115-147.
3. Biederman, I. and P.C. Gerhardstein, *Recognizing depth-rotated objects: Evidence and conditions for three-dimensional viewpoint invariance*. Journal of Experimental Psychology: Human Perception and Performance, 1993. **19**: p. 1162-1182.
4. Tarr, M.J. and H.H. Bülthoff, *Is human object recognition better described by geon structural descriptions or by multiple views? Comment on Biederman and Gerhardstein (1993)*. Journal of Experimental Psychology: Human Perception and Performance, 1995. **21**: p. 1494-1505.
5. Farah, M.J., *Is an object an object an object? Cognitive and neuropsychological investigations of domain-specificity in visual object recognition*. Current Directions in Psychological Science, 1992. **1**: p. 164-169.
6. Kanwisher, N., M.M. Chun, and P. ledden, *Functional imaging of human visual recognition*. Cognitive Brain Research, 1996. **5**: p. 55-67.
7. Kanwisher, N., J. McDermott, and M.M. Chun, *The fusiform face area: A module in human extra-striate cortex specialized for face perception*. Journal of Neuroscience, 1997. **17**: p. 1-10.
8. Kanwisher, N., *et al.*, *A locus in human extrastriate cortex for visual shape analysis*. Journal of Cognitive Neuroscience, 1997. **9**: p. 133-142.
9. Ishai, A., *et al.*, *fMRI reveals differential activation in the ventral object recognition pathway during the perception of faces, hourses and chairs*. Neuroimage, 1997. **5**(149).
10. Edelman, S., *Features of Recognition*. 1991, Rehovot, Isreal: Weizmann Institute of Science.
11. Jolicoeur, P., *Identification of disoriented objects: A dual-system theory*. Memory & Cognition, 1990. **13**: p. 289-303.
12. Tjan, B.S. and G.E. Legge, *The viewpoint complexity of an object recognition task*. Vision Research, 1998. **38**: p. 2335-50.
13. Jolicoeur, P., M.A. Gluck, and S.M. Kosslyn, *From pictures to words: Making the connection*. Cognitive Psychology, 1984. **16**: p. 243-275.
14. Gauthier, I., *et al.*, *Activation of the middle fusiform "face area" increases with expertise in recognizing novel objects*. Nature Neuroscience, 1999. **2**(6): p. 568-573.
15. Farah, M.J., *Visual Agnosia: Disorders of Object Recognition and What They Tell Us about Normal Vision*. 1990, Cambridge, MA: MIT Press.
